# PATTERN CLASS DEGENERACY IN AN UNRESTRICTED STORAGE DENSITY MEMORY

Christopher L. Scofield, Douglas L. Reilly,
Charles Elbaum, Leon N. Cooper

Nestor, Inc., 1 Richmond Square, Providence, Rhode Island, 02906.

## ABSTRACT

The study of distributed memory systems has produced a number of models which work well in limited domains. However, until recently, the application of such systems to real-world problems has been difficult because of storage limitations, and their inherent architectural (and for serial simulation, computational) complexity. Recent development of memories with unrestricted storage capacity and economical feedforward architectures has opened the way to the application of such systems to complex pattern recognition problems. However, such problems are sometimes underspecified by the features which describe the environment, and thus a significant portion of the pattern environment is often non-separable. We will review current work on high density memory systems and their network implementations. We will discuss a general learning algorithm for such high density memories and review its application to separable point sets. Finally, we will introduce an extension of this method for learning the probability distributions of non-separable point sets.

## INTRODUCTION

Information storage in distributed content addressable memories has long been the topic of intense study. Early research focused on the development of correlation matrix memories [1, 2, 3, 4]. Workers in the field found that memories of this sort allowed storage of a number of distinct memories no larger than the number of dimensions of the input space. Further storage beyond this number caused the system to give an incorrect output for a memorized input.

Recent work on distributed memory systems has focused on single layer, recurrent networks. Hopfield [5, 6] introduced a method for the analysis of settling of activity in recurrent networks. This method defined the network as a dynamical system for which a global function called the 'energy' (actually a Liapunov function for the autonomous system describing the state of the network) could be defined. Hopfield showed that flow in state space is always toward the fixed points of the dynamical system if the matrix of recurrent connections satisfies certain conditions. With this property, Hopfield was able to define the fixed points as the sites of memories of network activity.

Like its forerunners, the Hopfield network is limited in storage capacity. Empirical study of the system found that for randomly chosen memories, storage capacity was limited to $m \leq 0.15N$, where $m$ is the number of memories that could be accurately recalled, and $N$ is the dimensionality of the network (this has since been improved to $m \leq N$, [7, 8]). The degradation of memory recall with increased storage density is directly related to the proliferation in the state space of unwanted local minima which serve as basins of flow.

## UNRESTRICTED STORAGE DENSITY MEMORIES

Bachman et al. [9] have studied another relaxation system similar in some respects to the Hopfield network. However, in contrast to Hopfield, they have focused on defining a dynamical system in which the locations of the minima are explicitly known.

In particular, they have chosen a system with a Liapunov function given by

$$E = -1/L \sum_j Q_j \, | \mu - x_j | ^{-L}, \qquad (1)$$

where E is the total 'energy' of the network, $\mu(0)$ is a vector describing the initial network activity caused by a test pattern, and $x_j$, the site of the $j^{th}$ memory, for m memories in $R^N$. L is a parameter related to the network size. Then $\mu(0)$ relaxes to $\mu(T) = x_j$ for some memory j according to

$$\dot{\mu} = -\sum_j Q_j \, | \, \mu - x_j \, |^{-(L+2)} (\mu - x_j) \qquad (2)$$

This system is isomorphic to the classical electrostatic potential between a positive (unit) test charge, and negative charges $Q_j$ at the sites $x_j$ (for a 3-dimensional input space, and $L = 1$). The N-dimensional Coulomb energy function then defines exactly m basins of attraction to the fixed points located at the charge sites $x_j$. It can been shown that convergence to the closest distinct memory is guaranteed, *independent of the number of stored memories m,* for proper choice of N and L [9, 10].

Equation 1 shows that each cell receives feedback from the network in the form of a scalar

$$\sum_j Q_j \, | \, \mu - x_j \, |^{-L} . \qquad (3)$$

Importantly, this quantity is the same for all cells; it is as if a single virtual cell was computing the distance in activity space between the current state and stored states. The result of the computation is then broadcast to all of the cells in the network. A 2-layer feedforward network implementing such a system has been described elsewhere[10].

The connectivity for this architecture is of order m·N, where m is the number of stored memories and N is the dimensionality of layer 1. This is significant since the addition of a new memory m' = m + 1 will change the connectivity by the addition of N + 1 connections, whereas in the Hopfield network, addition of a new memory requires the addition of 2N + 1 connections.

An equilibrium feedforward network with similar properties has been under investigation for some time [11]. This model does not employ a relaxation procedure, and thus was not originally framed in the language of Liapunov functions. However, it is possible to define a similar system if we identify the locations of the 'prototypes' of this model as the locations in state space of potentials which satisfy the following conditions

$$E_j = -Q_j / R_o \quad \text{for} \, | \, \mu - x_j \, | < \lambda_j \qquad (4)$$

$$= 0 \qquad \text{for} \, | \, \mu - x_j \, | > \lambda_j.$$

where $R_o$ is a constant.

This form of potential is often referred to as the 'square-well' potential. This potential may be viewed as a limit of the N-dimensional Coulomb potential, in which the $1/R$ ($L = 1$) well is replaced with a square well (for which $L >> 1$). Equation 4 describes an energy landscape which consists of plateaus of zero potential outside of wells with flat, zero slope basins. Since the landscape has only flat regions separated by discontinuous boundaries, the state of the network is always at equilibrium, and relaxation does not occur. For this reason, this system has been called an equilibrium model. This model, also referred to as the Restricted Coulomb Energy (RCE)[14] model, shares the property of unrestricted storage density.

## LEARNING IN HIGH DENSITY MEMORIES

A simple learning algorithm for the placement of the wells has been described in detail elsewhere [11, 12].

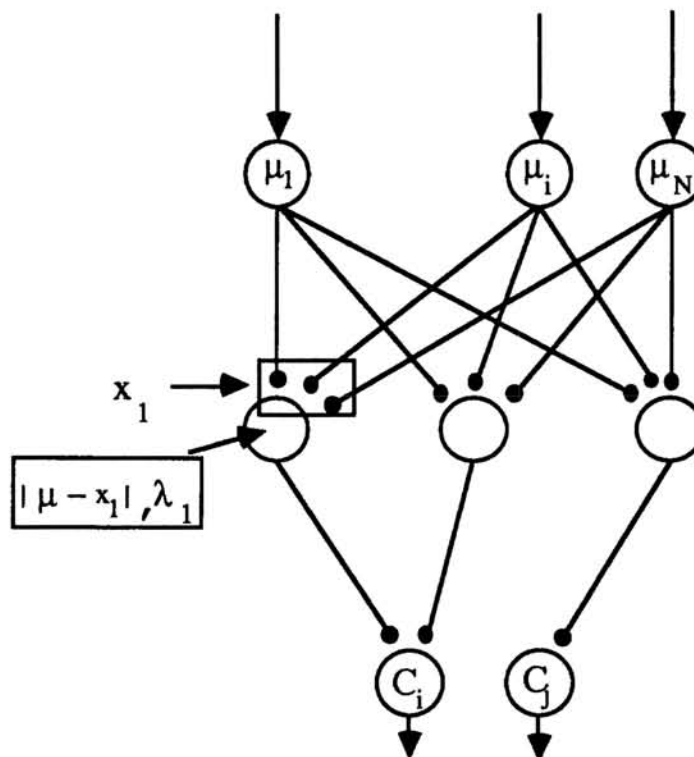

Figure1: 3-layer feedforward network. Cell i computes the quantity $|\mu - x_i|$ and compares to internal threshold $\lambda_i$.

Reilly et. al. have employed a three layer feedforward network (figure 1) which allows the generalization of a content addressable memory to a pattern classification memory. Because the locations of the minima are explicitly known in the equilibrium model, it is possible to dynamically program the energy function for an arbitrary energy landscape. This allows the construction of geographies of basins associated with the classes constituting the pattern environment. Rapid learning of complex, non-linear, disjoint, class regions is possible by this method [12, 13].

## LEARNING NON-SEPARABLE CLASS REGIONS

Previous studies have focused on the acquisition of the geography and boundaries of non-linearly separable point sets. However, a method by which such high density models can acquire the probability distributions of non-separable sets has not been described.

Non-separable sets are defined as point sets in the state space of a system which are labelled with multiple class affiliations. This can occur because the input space has not carried all of the features in the pattern environment, or because the pattern set itself is not separable. Points may be degenerate with respect to the explicit features of the space, however they may have different probability distributions within the environment. This structure in the environment is important information for the identification of patterns by such memories in the presence of feature space degeneracies.

We now describe one possible mechanism for the acquisition of the probability distribution of non-separable points. It is assumed that all points in some region R of the state space of the network are the site of events $\mu(0, C_i)$ which are examples of pattern classes $C = \{C_1,..., C_M\}$. A basin of attraction, $x_k(C_i)$, defined by equation 4, is placed at each site $\mu(0, C_i)$ unless

$$| \mu(0, C_i) - x_j(C_i) | < R_o, \qquad (5)$$

that is, unless a memory at $x_j$ (of the class $C_i$) already contains $\mu(0, C_i)$. The initial values of $Q_o$ and $R_o$ at $x_k(C_i)$ are a constant for all sites $x_j$. Thus as events of the classes $C_1,...,C_M$ occur at a particular site in R, multiple wells are placed at this location.

If a well $x_j(C_i)$ correctly covers an event $\mu(0, C_i)$, then the charge at that site (which defines the depth of the well) is incremented by a constant amount $\Delta Q_0$. In this manner, the region R is covered with wells of all classes $\{C_1,..., C_M\}$, with the depth of well $x_j(C_i)$ proportional to the frequency of occurence of $C_i$ at $x_j$.

The architecture of this network is exactly the same as that already described.  As before, this network acquires a new cell for each well placed in the energy landscape.  Thus we are able to describe the meaning of wells that overlap as the competition by multiple cells in layer 2 in firing for the pattern of activity in the input layer.

## APPLICATIONS

This system has been applied to a problem in the area of risk assessment in mortgage lending.  The input space consisted of feature detectors with continuous firing rates proportional to the values of 23 variables in the application for a mortgage.  For this set of features, a significant portion of the space was non-separable.

Figures 2a and 2b illustrate the probability distributions of high and low risk applications for two of the features.  It is clear that in this 2-dimensional subspace, the regions of high and low risk are non-separable but have different distributions.

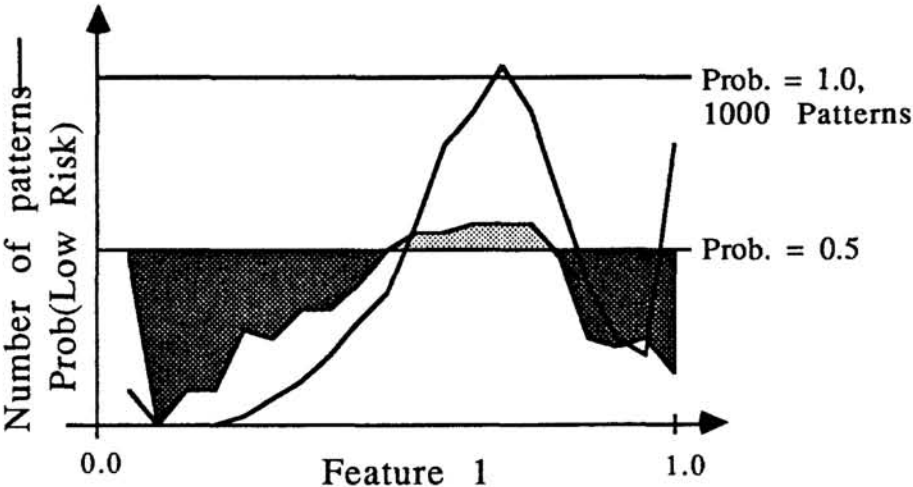

Figure 2a: Probability distribution for High and Low risk patterns for feature 1.

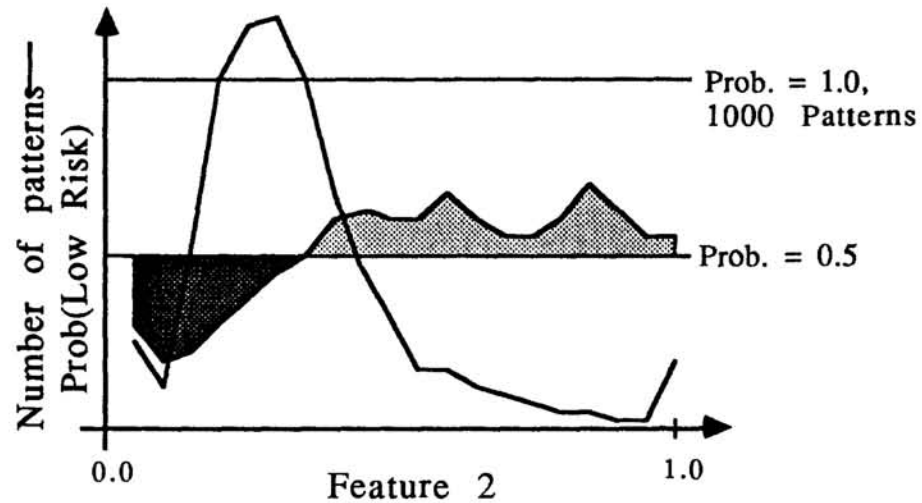

Figure 2b: Probability distribution for High and Low risk patterns for feature 2.

Figure 3 depicts the probability distributions acquired by the system for this 2-dimensional subspace. In this image, circle radius is proportional to the degree of risk: Small circles are regions of low risk, and large circles are regions of high risk.

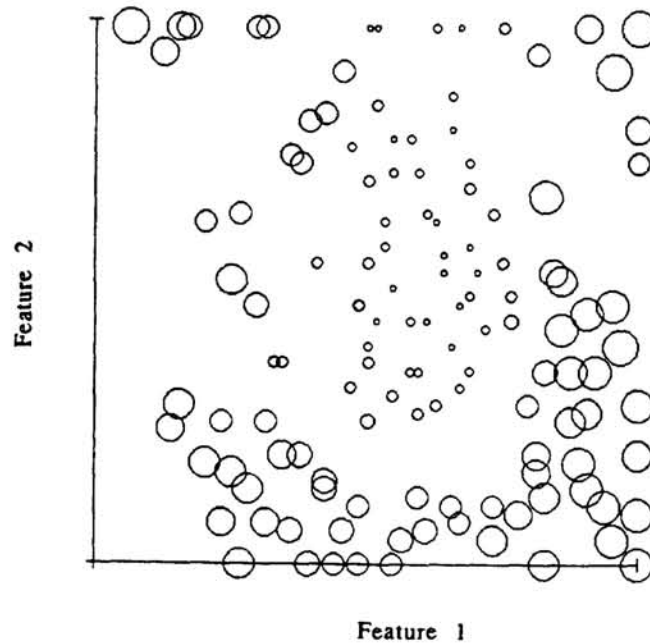

Figure 3: Probability distribition for Low and High risk. Small circles indicate low risk regions and large circles indicate high risk regions.

Of particular interest is the clear clustering of high and low risk regions in the 2-d map. Note that the regions are in fact non-linearly separable.

## DISCUSSION

We have presented a simple method for the acquisition of probability distributions in non-separable point sets. This method generates an energy landscape of potential wells with depths that are proportional to the local probability density of the classes of patterns in the environment. These well depths set the probability of firing of class cells in a 3-layer feedforward network.

Application of this method to a problem in risk assessment has shown that even completely non-separable subspaces may be modeled with surprising accuracy. This method improves pattern classification in such problems with little additional computational burden.

This algorithm has been run in conjunction with the method described by Reilly et. al.[11] for separable regions. This combined system is able to generate non-linear decision surfaces between the separable zones, and approximate the probability distributions of the non-separable zones in a seemless manner. Further discussion of this system will appear in future reports.

Current work is focused on the development of a more general method for modelling the scale of variations in the distributions. Sensitivity to this scale suggests that the transition from separable to non-separable regions is smooth and should not be handled with a 'hard' threshold.

## ACKNOWLEDGEMENTS

We would like to thank Ed Collins and Sushmito Ghosh for their significant contributions to this work through the development of the mortgage risk assessment application.

## REFERENCES

[1] Anderson, J.A.: A simple neural network generating an interactive memory. Math. Biosci. **14**, 197-220 (1972).

[2] Cooper, L.N.: A possible organization of animal memory and learning. In: Proceedings of the Nobel Symposium on Collective Properties of Physical Systems, Lundquist, B., Lundquist, S. (eds.). (24), 252-264 London, New York: Academic Press 1973.

[3] Kohonen, T.: Correlation matrix memories. IEEE Trans. Comput. 21, 353-359 (1972).

[4] Kohonen, T.: Associative memory - a system-theoretical approach. Berlin, Heidelberg, New York: Springer 1977.

[5] Hopfield, J.J.: Neural networks and physical systems with emergent collective computational abilities. Proc. Natl. Acad. Sci. USA 79, 2554-2558 (April 1982).

[6] Hopfield, J.J.: Neurons with graded response have collective computational properties like those of two-state neurons. Proc. Natl. Acad. Sci. USA 81, 2088-3092 (May, 1984).

[7] Hopfield, J.J., Feinstein, D.I., Palmer, R.G.: 'Unlearning' has a stabilizing effect in collective memories. Nature 304, 158-159 (July 1983).

[8] Potter, T.W.: Ph.D. Dissertation in advanced technology, S.U.N.Y. Binghampton, (unpublished).

[9] Bachmann, C.M., Cooper, L.N., Dembo, A., Zeitouni, O.: A relaxation model for memory with high density storage. to be published in Proc. Natl. Acad. Sci. USA.

[10] Dembo, A., Zeitouni, O.: ARO Technical Report, Brown University, Center for Neural Science, Providence, R.I., (1987), also submitted to Phys. Rev. A.

[11] Reilly, D.L., Cooper, L.N., Elbaum, C.: A neural model for category learning. Biol. Cybern. 45, 35-41 (1982).

[12] Reilly, D.L., Scofield, C., Elbaum, C., Cooper, L.N.: Learning system architectures composed of multiple learning modules. to appear in Proc. First Int'l. Conf. on Neural Networks (1987).

[13] Rimey, R., Gouin, P., Scofield, C., Reilly, D.L.: Real-time 3-D object classification using a learning system. Intelligent Robots and Computer Vision, Proc. SPIE 726 (1986).

[14] Reilly, D.L., Scofield, C. L., Elbaum, C., Cooper, L.N: Neural Networks with low connectivity and unrestricted memory storage density. To be published.
